# Partially labeled classification with Markov random walks

**Martin Szummer**
MIT AI Lab & CBCL
Cambridge, MA 02139
*szummer@ai.mit.edu*

**Tommi Jaakkola**
MIT AI Lab
Cambridge, MA 02139
*tommi@ai.mit.edu*

## Abstract

To classify a large number of unlabeled examples we combine a limited number of labeled examples with a Markov random walk representation over the unlabeled examples. The random walk representation exploits any low dimensional structure in the data in a robust, probabilistic manner. We develop and compare several estimation criteria/algorithms suited to this representation. This includes in particular multi-way classification with an average margin criterion which permits a closed form solution. The time scale of the random walk regularizes the representation and can be set through a margin-based criterion favoring unambiguous classification. We also extend this basic regularization by adapting time scales for individual examples. We demonstrate the approach on synthetic examples and on text classification problems.

## 1 Introduction

Classification with partially labeled examples involves a limited dataset of labeled examples as well as a large unlabeled dataset. The unlabeled examples to be classified provide information about the structure of the domain while the few labeled examples identify the classification task expressed in this structure. A common albeit tacit assumption in this context associates continuous high-density clusters in the data with pure classes. When this assumption is appropriate, we only require one labeled point for each cluster to properly classify the whole dataset.

Data points are typically given relative to a global coordinate system with an associated metric. While the metric may provide a reasonable local similarity measure, it is frequently inadequate as a measure of global similarity. For example, the data may lie on a submanifold of the space, revealed by the density, and any global comparisons should preferably be made along the manifold structure. Moreover, we often wish to assign higher similarity values to examples contained in the same high-density regions or clusters implying that comparisons ought to incorporate the density in addition to the manifold structure.

A representation of examples that satisfies these and other desiderata can be constructed through a Markov random walk similarly to [3]. The resulting global comparisons of examples integrate a "volume" of paths connecting the examples as opposed to shortest paths that are susceptible to noise. The time scale of the Markov process (the number of transitions) will permit us to incorporate the cluster structure in the data at different levels

of granularity. We start by defining the representation and subsequently develop several classification methods naturally operating on such representations.

## 2   Representation based on Markov random walks

We define a Markov random walk based on a locally appropriate metric [3]. The metric is the basis for the neighborhood graph, associated weights on the edges, and consequently the transition probabilities for the random walk. The new representation for the examples can be obtained naturally from the random walk.

More formally, consider a set of points $\{\mathbf{x}_1, \ldots, \mathbf{x}_N\}$ with a metric $d(\mathbf{x}_i, \mathbf{x}_j)$. We first construct a symmetrized $K$ nearest neighbor graph $G$ over the points and assign a weight $W_{ij} = \exp(-d(\mathbf{x}_i, \mathbf{x}_k)/\sigma)$ to each undirected edge in the graph. The weights are symmetric and $W_{ii} = 1$ as we include self-loops; $W_{ij} = 0$ for all non-neighbors. Note that the product of weights along a path in the graph relates to the total length of the path in the same way as the edge weights relate to the distances between the corresponding points. The one-step transition probabilities $p_{ik}$ from $i$ to $k$ are obtained directly from these weights:

$$p_{ik} = \frac{W_{ik}}{\sum_j W_{ij}} \tag{1}$$

($p_{ik} = 0$ for any non-neighbor $k$). While the weights $W_{ik}$ are symmetric, the transition probabilities $p_{ik}$ generally are not, because the normalization varies from node to node.

We use $P_{t|0}(k|i)$ to denote the $t$ step transition probabilities ($t$ here should be interpreted as a parameter, not as a random variable). If we organize the one step transition probabilities as a matrix $\mathbf{A}$ whose $i, k$-th entry is $p_{ik}$, we can simply use a matrix power to calculate

$$P_{t|0}(k|i) = [\mathbf{A}^t]_{ik}. \tag{2}$$

The matrix $\mathbf{A}$ is row stochastic so that rows sum to 1.

We assume that the starting point for the Markov random walk is chosen uniformly at random, i.e., $P(i) = 1/N$. We can now evaluate the probability that the Markov process started from $i$ given that it ended up in $k$ after $t$ steps. These conditional probabilities $P_{0|t}(i|k)$ define our new representation for the examples. In other words, each point $k$ is associated with a vector of conditional probabilities $P_{0|t}(i|k)$, $i = 1, \ldots, N$. The points in this representation are close whenever they have nearly the same distribution over the starting states. This representation is crucially affected by the time scale parameter $t$. When $t \to \infty$, all the points become indistinguishable provided that the original neighborhood graph is connected. Small values of $t$, on the other hand, merge points in small clusters. In this representation $t$ controls the resolution at which we look at the data points (cf [3]).

The representation is also influenced by $K$, $\sigma$, and the local distance metric $d$, which together define the one-step transition probabilities (see section 4).

## 3   Parameter estimation for classification

Given a partially labeled data set $\{(\mathbf{x}_1, \tilde{y}_1), \ldots, (\mathbf{x}_L, \tilde{y}_L), \mathbf{x}_{L+1}, \ldots, \mathbf{x}_N\}$, we wish to classify the unlabeled points. The labels may come from two or more classes, and typically, the number of labeled points $L$ is a small fraction of the total points $N$.

Our classification model assumes that each data point has a label or a distribution $P(y|i)$ over the class labels. These distributions are unknown and represent the parameters to be estimated. Now given a point $k$, which may be labeled or unlabeled, we interpret the point

as a sample from the $t$ step Markov random walk. Since labels are associated with the original (starting) points, the posterior probability of the label for point $k$ is given by

$$P_{\text{post}}(y|k) = \sum_i P(y|i)P_{0|t}(i|k). \tag{3}$$

To classify the $k$-th point, we choose the class that maximizes the posterior:
$c_k = \text{argmax}_c \, P_{\text{post}}(y = c|k)$.

We will now discuss two techniques for estimating the unknown parameters $P(y|i)$: maximum likelihood with EM, and maximum margin subject to constraints.

### 3.1 EM estimation

The estimation criterion here is the conditional log-likelihood of the labeled points

$$\sum_{k=1}^{L} \log P(\tilde{y}_k|k) = \sum_{k=1}^{L} \log \sum_{i=1}^{N} P(\tilde{y}_k|i)P_{0|t}(i|k). \tag{4}$$

Since $P_{0|t}(i|k)$ are fixed for any specific $t$, this objective function is jointly concave in the free parameters and has a unique maximum value. The concavity also guarantees that this optimization is easily performed via the EM algorithm.

Let $P(i|k, \tilde{y}_k)$ be the soft assignment for component $i$ given $(k, \tilde{y}_k)$, i.e.,
$P(i|k, \tilde{y}_k) \propto P(\tilde{y}_k|i)P_{0|t}(i|k)$. The EM algorithm iterates between the E-step, where $P(i|k, \tilde{y}_k)$ are recomputed from the current estimates of $P(y|i)$, and the M-step where we update $P(y|i) \leftarrow \sum_{k:\tilde{y}_k=y} P(i|k, \tilde{y}_k)/\sum_k P(i|k, \tilde{y}_k)$, (see [1]).

### 3.2 Margin based estimation

An alternative discriminative formulation is also possible, one that is more sensitive to individual classification decisions rather than the product of their likelihoods. Define the margin of the classifier on labeled point $k$ and class $d$ to be $\gamma_{kd} = P_{\text{post}}(y = \tilde{y}_k|k) - P_{\text{post}}(y = d|k)$. For correct classification, the margin should be nonnegative for all classes $d$ other than $\tilde{y}_k$, i.e. $\gamma_{kd} \geq 0$, and be zero for the correct class $\gamma_{k\,\tilde{y}_k} = 0$.

During training, find the parameters $P(y|i)$ that maximize the average margin on the labeled points, thereby forcing most of them to be correctly classified. Unbalanced classes are handled by the per class margin, and we obtain the linear program

$$\max_{P(y|i),\gamma_{kd}} \frac{1}{C(C-1)} \sum_{k=1}^{L} \sum_{d=1}^{C} \frac{1}{N_{C(k)}} \gamma_{kd} \qquad \text{subject to} \tag{5}$$

$$P_{\text{post}}(y = \tilde{y}_k|k) \geq P_{\text{post}}(y = d|k) + \gamma_{kd} \quad \forall k \in 1 \ldots L \, \forall d \in 1 \ldots C \tag{6}$$

$$\sum_{c=1}^{C} P(y = c|i) = 1 \text{ and } 0 \leq P(y|i) \leq 1 \qquad \forall i. \tag{7}$$

Here $C$ denotes the number of classes and $N_{C(k)}$ gives the number of labeled points in the same class as $k$. The solution is achieved at extremal points of the parameter set and thus it is not surprising that the optimal parameters $P(y|i)$ reduce to hard values (0 or 1). The solution to this linear program can be found in closed form:

$$P(y = c_i|i) = \left\{ \begin{array}{ll} 1 & \text{if } c_i = \text{argmax}_c \frac{1}{N_c} \sum_{k:\tilde{y}_k=c} P_{0|t}(i|k) \\ 0 & \text{otherwise.} \end{array} \right. \tag{8}$$

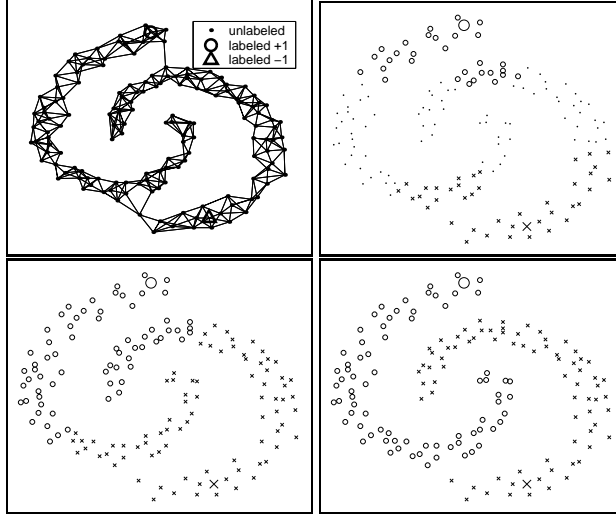

Figure 1: Top left: local connectivity for $K$=5 neighbors. Below are classifications using Markov random walks for $t$=3, 10, and 30 (top to bottom, left to right), estimated with average margin. There are two labeled points (large cross, triangle) and 148 unlabeled points, classified (small crosses, triangles) or unclassified (small dots).

The resulting posterior probabilities can also be written compactly as $P_{\text{post}}(y = c|k) = \sum_{i:P(y=c|i)=1} P_{0|t}(i|k)$. The closed form solution for the label distributions facilitates an easy cross-validated setting of the various parameters involved in the example representations.

The large margin restricts the $V_\gamma$ dimension of the classifier (section 3.4) and encourages generalization to correct classification of the unlabeled points as well. Note that the margins are bounded and have magnitude less than 1, reducing the risk that any single point would dominate the average margin. Moreover, this criterion maximizes a sum of probabilities, whereas likelihood maximizes a product of probabilities, which is easily dominated by low probability outliers.

Other margin-based formulations are also possible. For separable problems, we can maximize the minimum margin instead of the average margin. In the case of only two classes, we then have only one global margin parameter $\gamma$ for all labeled points. The algorithm focuses all its attention at the site of the minimum margin, which unfortunately could be an outlier. If we tackled noisy or non-separable problems by adding a linear slack variable to each constraint, we would arrive at the average margin criterion given above (because of linearity).

Average- and min-margin training yields hard parameters 0 or 1. The risk of overfitting is controlled by the smooth representation and can be regularized by increasing the time parameter $t$. If further regularization is desired, we have also applied the maximum entropy discrimination framework [2, 1] to bias the solution towards more uniform values. This additional regularization has resulted in similar classification performance but adds to the computational cost.

### 3.3 Examples

Consider an example (figure 1) of classification with Markov random walks. We are given 2 labeled and 148 unlabeled points in an intertwining two moons pattern. This pattern has a

manifold structure where distances are locally but not globally Euclidean, due to the curved arms. Therefore, the pattern is difficult to classify for traditional algorithms using global metrics, such as SVM. We use a Euclidean local metric, $K=5$ and $\sigma=0.6$ (the box has extent $2 \times 2$), and show three different timescales. At $t=3$ the random walk has not connected all unlabeled points to some labeled point. The parameters for unconnected points do not affect likelihood or margin, so we assign them uniformly to both classes. The other points have a path to only one of the classes, and are therefore fully assigned to that class. At $t=10$ all points have paths to labeled points but the Markov process has not mixed well. Some paths do not follow the curved high-density structure, and instead cross between the two clusters. When the Markov process is well-mixed at $t=30$, the points are appropriately labeled. The parameter assignments are hard, but the class posteriors are weighted averages and remain soft.

### 3.4   Sample size requirements

Here we quantify the sample size that is needed for accurate estimation of the labels for the unlabeled examples. Since we are considering a transduction problem, i.e., finding labels for already observed examples, the sample size requirements can be assessed directly in terms of the representation matrix. As before, the probabilities $P_{0|t}(i|j)$ and $P_{0|t}(i|k)$ denote the conditional probabilities of having started the random walk in $i$ given that the process ends up in $j$, $k$, respectively. For simplicity, we consider a binary problem with classes 1 and -1, and let $w_i = P(y = 1|i) - P(y = -1|i)$. Classification decisions are then directly based on the sign of $f(k) = \sum_{i=1}^{N} w_i P_{0|t}(i|k)$.

**Lemma 1** *Consider the absolute distance between the representations of two points $d_{jk} = \sum_{i=1}^{N} |P_{0|t}(i|j) - P_{0|t}(i|k)|$. The $V_\gamma$ dimension [5] of the binary transductive classifier $f(k)$ is upper bounded by the number of connected components of a graph with $N$ nodes and adjacency matrix $\mathbf{A}$, where $\mathbf{A}_{jk} = 1$ if $d_{jk} \leq \gamma$ and zero otherwise.*

**Proof:** To evaluate $V_\gamma$, a measure of the capacity of the classifier, we count the number of complete labelings $y_k$ consistent with the margin constraints $y_k f(k) \geq \gamma$ for all $k$ (labeled and unlabeled points). First, we establish that all examples $j$ and $k$ for which $d_{jk} \leq \gamma$ must have the same label. This follows directly from

$$|f(j) - f(k)| \leq \sum_{i=1}^{N} |P_{0|t}(i|j) - P_{0|t}(i|k)| \, |P_{\text{post}}(y = 1|k) - P_{\text{post}}(y = -1|k)| \quad (9)$$

$$\leq \sum_{i=1}^{N} |P_{0|t}(i|j) - P_{0|t}(i|k)| = d_{jk}, \quad (10)$$

as this difference must be larger than $\gamma$ for the discriminant functions to have different signs. Since any pair of examples for which $d_{jk} \leq \gamma$ share the same label, different labels can be assigned only to examples not connected by the $d_{jk} \leq \gamma$ relation, i.e., examples in distinct connected components.$\square$

This theorem applies more generally to any transductive classifier based on a weighted representation of examples so long as the weights are bounded in $[-1, 1]$.

To determine the sample size needed for a given dataset, and a desired classification margin $\gamma$, let $r = V_\gamma$ dimension. With high probability we can correctly classify the unlabeled points given $O(r \log r)$ labeled examples [4]. This can also be helpful to determine timescale $t$ since it is reflected in the $V_\gamma$, for example $V_\gamma = N$ for $t=0$ and $V_\gamma = 1$ for $t=\infty$ for the full range of $\gamma \in [0, 2]$.

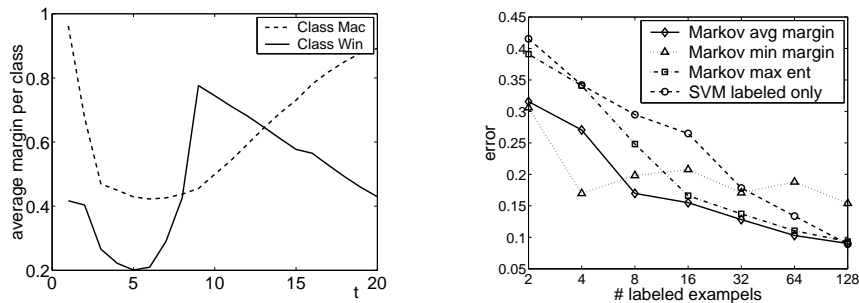

Figure 2: Windows vs. Mac text data. Left: Average per class margins for different $t$, 16 labeled documents. Right: Classification accuracy, between 2 and 128 labeled documents, for Markov random walks and best SVM.

# 4   Choices for $d$, $K$, $\sigma$, and $t$

The classifier is robust to rough heuristic choices of $d(\cdot, \cdot)$, $K$, and $\sigma$, as follows. The local similarity measure $d(\cdot, \cdot)$ is typically given (Euclidean distance). The local neighborhood size $K$ should be on the order of the manifold dimensionality, sufficiently small to avoid introducing edges in the neighborhood graph that span outside the manifold. However, $K$ must be large enough to preserve local topology, and ideally large enough to create a singly connected graph, yielding an ergodic Markov process. The local scale parameter $\sigma$ trades off the emphasis on shortest paths (low $\sigma$ effectively ignores distant points), versus volume of paths (high $\sigma$).

The smoothness of the random walk representation depends on $t$, the number of transitions. This is a regularization parameter akin to the kernel width of a density estimator. In the limiting case $t=1$, we employ only the local neighborhood graph. As a special case, we obtain the kernel expansion representation [1] by $t=1$, $K=N$, and squared Euclidean distance. If all points are labeled, we obtain the $K$-nearest neighbors classifier by $t=1$, $\sigma \rightarrow \infty$. In the limiting case $t=\infty$ the representation for each node becomes a flat distribution over the points in the same connected component.

We can choose $t$ based on a few unsupervised heuristics, such as the mixing time to reach the stationary distribution, or dissipation of mutual information [3].

However, appropriate $t$ depends on the classification task. For example, if classes change quickly over small distances, we want a sharper representation given by smaller $t$. Cross-validation could provide a supervised choice of $t$ but requires too many labeled points for good accuracy. Instead, we propose to choose $t$ that maximizes the average margin per class, on both labeled and unlabeled data. Plot $\frac{1}{N_c} \sum_{k:\text{class}(k)=c} \sum_d \gamma_{kd}$ for each c, separately for labeled and unlabeled points to avoid issues of their relative weights. For labeled points, $\text{class}(k) = \tilde{y}_k$, for unlabeled points, $\text{class}(k)$ is the class assigned by the classifier. Figure 2 shows the average margin as a function of $t$, for a large text dataset (section 5). We want large margins for both classes simultaneously, so $t = 8$ is a good choice, and also gave the best cross-validation accuracy.

## 4.1   Adaptive time scales

So far, we have employed a single global value of $t$. However, the desired smoothness may be different at different locations (akin to adaptive kernel widths in kernel density estimation). At the simplest, if the graph has multiple connected components, we can

set individual $t$ for each component. Ideally, each point has its own time scale, and the choice of time scale is optimized jointly with the classifier parameters. Here we propose a restricted version of this criterion where we find individual time scales $t_k$ for each unlabeled point but estimate a single timescale for labeled points as before.

The principle by which we select the time scales for the unlabeled points encourages the node identities to become the only common correlates for the labels. More precisely, define $P(y|k)$ for any unlabeled point $k$ as

$$P(y|k) = \frac{1}{Z_k} \sum_{i:\tilde{y}_i=y} P_{0|t_k}(i|k), \tag{11}$$

where $Z_k = \sum_i P_{0|t_k}(i|k)$ and both summations are only over the labeled points. Moreover, let $P(y)$ be the overall probability over the labels across the unlabeled points or

$$P(y) = \sum_k P(k)P(y|k), \tag{12}$$

where $P(k)$ is uniform over the unlabeled points, corresponding to the start distribution. Note that $P(y)$ remains a function of all the individual time scales for the unlabeled points. With these definitions, the principle for setting the time scales reduces to maximizing the mutual information between the label and the node identity:

$$\{t_1, \ldots, t_m\} = \arg\max_{t_1,\ldots,t_m} I(y;k) = \arg\max_{t_1,\ldots,t_m} \{H(y) - \sum_j P(k=j)H(y|k=j)\}. \tag{13}$$

$H(y)$ and $H(y|k)$ are the marginal and conditional entropies over the labels and are computed on the basis of $P(y)$ and $P(y|k)$, respectively. Note that the ideal setting of the time scales would be one that determines the labels for the unlabeled points uniquely on the basis of only the labeled examples while at the same time preserving the overall variability of the labels across the nodes. This would happen, for example, if the labeled examples fall on distinct connected components. We optimize the criterion by an axis parallel search, trying only discrete values of $t_k$ large enough that at least one labeled point is reached from each unlabeled point. We initialize $t_k$ to the smallest number of transitions needed to reach a labeled point. Empirically we have found that this initialization is close to the refined solution given by the objective. The objective is not concave, but separate random initializations generally yield the same answer, and convergence is rapid requiring about 5 iterations.

## 5 Experimental results

We applied the Markov random walk approach to partially labeled text classification, with few labeled documents but many unlabeled ones. Text documents are represented by high-dimensional vectors but only occupy low-dimensional manifolds, so we expect Markov random walk to be beneficial. We used the `mac` and `windows` subsets from the 20 news-groups dataset[1]. There were 958 and 961 examples in the two classes, with 7511 dimensions. We estimated the manifold dimensionality to exceed 7, and a histogram of the distances to the 10 nearest neighbor is peaked at 1.3. We chose a Euclidean local metric, $K$=10, which leads to a single connected component, and $\sigma$=0.6 for a reasonable falloff. The average margin criterion indicated $t = 8$, and we also cross-validated and plotted the decay of mutual information over $t$. We trained both the EM and the margin-based formulations, using between 2 and 128 labeled points, treating all remaining points as unlabeled. We trained on 20 random splits balanced for class labels, and tested on a fixed separate set of 987 points. Results in figure 2 show that Markov random walk based algorithms have

a clear advantage over the best SVM using only labeled data (which had a linear kernel and $C$=3), out of linear and Gaussian kernels, different kernel widths and values of $C$. The advantage is especially noticeable for few labeled points, but decreases thereafter. The average margin classifier performs best overall. It can handle outliers and mislabeled points, unlike the maximum min margin classifier that stops improving once 8 or more labeled points are supplied.

The adaptive timescale criterion favors relatively small timescales for this dataset. For 90% of the unlabeled points, it picks the smallest timescale that reaches a labeled point, which is at most 8 for any point. As the number of labeled points increases, shorter times are chosen. For a few points, the criterion picks a maximally smooth representation (the highest timescale considered here, $t$=12), possibly to increase the $H(y)$ criterion. However, our preliminary experiments suggest that the adaptive time scales do not have a special classification advantage for this dataset.

# 6 Discussion

The Markov random walk representation of examples provides a robust variable resolution approach to classifying data sets with significant manifold structure and very few labels. The average margin estimation criterion proposed in this context leads to a closed form solution and strong empirical performance. When the manifold structure is absent or unrelated to the classification task, however, our method cannot be expected to derive any particular advantage.

There are a number of possible extensions of this work. For example, instead of choosing a single overall resolution or time scale $t$, we may combine multiple choices. This can be done either by maintaining a few choices explicitly or including all time scales in a parametric form as in $e^{\mathbf{A}t} = I + t\mathbf{A} + t^2\mathbf{A}^2/2! + \dots$ [7], but it is unclear whether the exponential decay is desirable. To facilitate continuum limit analysis (and establish better correspondence with the underlying density), we can construct the neighborhood graph on the basis of $\epsilon$-balls rather than $K$ nearest neighbors.

**Acknowledgements**

The authors gratefully acknowledge support from Nippon Telegraph & Telephone (NTT) and NSF ITR grant IIS-0085836.

## Footnotes

[1] Processed as 20news-18827, `http://www.ai.mit.edu/~jrennie/20Newsgroups/`, removing rare words, duplicate documents, and performing tf-idf mapping.

# References

[1] Szummer, M; Jaakkola, T. (2000) Kernel expansions with unlabeled examples. NIPS 13.

[2] Jaakkola, T; Meila, M; Jebara, T. (1999) Maximum entropy discrimination. NIPS 12.

[3] Tishby, N; Slonim, N. (2000) Data clustering by Markovian relaxation and the Information Bottleneck Method. NIPS 13.

[4] Blum, A; Chawla, S. (2001) Learning from Labeled and Unlabeled Data using Graph Mincuts. ICML.

[5] Alon, N. et al (1997) Scale-sensitive Dimensions, Uniform Convergence, and Learnability. *J. ACM*, **44 (4)** 615-631

[6] Tenenbaum, J; de Silva, V; Langford J. (2000) A Global Geometric Framework for Nonlinear Dimensionality Reduction. *Science* **290 (5500)**: 2319-2323.

[7] Kondor, I; Lafferty J; (2001) Diffusion kernels in continuous spaces. Tech report CMU, to appear.
